# Nonparametric Greedy Algorithms for the Sparse Learning Problem

**Han Liu** and **Xi Chen**

School of Computer Science
Carnegie Mellon University
Pittsburgh, PA 15213

## Abstract

This paper studies the forward greedy strategy in sparse nonparametric regression. For additive models, we propose an algorithm called additive forward regression; for general multivariate models, we propose an algorithm called generalized forward regression. Both algorithms simultaneously conduct estimation and variable selection in nonparametric settings for the high dimensional sparse learning problem. Our main emphasis is empirical: on both simulated and real data, these two simple greedy methods can clearly outperform several state-of-the-art competitors, including LASSO, a nonparametric version of LASSO called the sparse additive model (SpAM) and a recently proposed adaptive parametric forward-backward algorithm called Foba. We also provide some theoretical justifications of specific versions of the additive forward regression.

## 1  Introduction

The linear model is a mainstay of statistical inference. At present, there are two major approaches to fit sparse linear models: *convex regularization* and *greedy pursuit*. The convex regularization approach regularizes the model by adding a sparsity constraint, leading to methods like LASSO [19, 7] or the Dantzig selector [6]. The greedy pursuit approach regularizes the model by iteratively selecting the current optimal approximation according to some criteria, leading to methods like the matching pursuit [14] or orthogonal matching pursuit (OMP) [20].

Substantial progress has been made recently on applying the convex regularization idea to fit sparse additive models. For splines, Lin and Zhang [12] propose a method called COSSO, which uses the sum of reproducing kernel Hilbert space norms as a sparsity inducing penalty, and can simultaneously conduct estimation and variable selection; Ravikumar et al. [17, 16] develop a method called SpAM. The population version of SpAM can be viewed as a least squares problem penalized by the sum of $L_2(P)$-norms; Meier et al. [15] develop a similar method using a different sparsity-smoothness penalty, which guarantees the solution to be a spline. All these methods can be viewed as different nonparametric variants of LASSO. They have similar drawbacks: (i) it is hard to extend them to handle general multivariate regression where the mean functions are no longer additive; (ii) due to the large bias induced by the regularization penalty, the model estimation is suboptimal. One way to avoid this is to resort to two-stage procedures as in [13], but the method becomes less robust due to the inclusion of an extra tuning parameter in the first stage.

In contrast to the convex regularization methods, the greedy pursuit approaches do not suffer from such problems. Instead of trying to formulate the whole learning task to be a global convex optimization, the greedy pursuit approaches adopt iterative algorithms with a local view. During each iteration, only a small number of variables are actually involved in the model fitting so that the whole inference only involves low dimensional models. Thus they naturally extend to the general multivariate regression and do not induce large estimation bias, which makes them especially suitable for high dimensional nonparametric inference. However, the greedy pursuit approaches do not attract as

much attention as the convex regularization approaches in the nonparametric literature. For additive models, the only work we know of are the sparse boosting [4] and multivariate adaptive regression splines (MARS) [9]. These methods mainly target on additive models or lower-order functional ANOVA models, but without much theoretical analysis. For general multivariate regression, the only available method we are aware of is rodeo [11]. However, rodeo requires the total number of variables to be no larger than a double-logarithmic of the data sample size, and does not explicitly identify relevant variables.

In this paper, we propose two new greedy algorithms for sparse nonparametric learning in high dimensions. By extending the idea of the orthogonal matching pursuit to nonparametric settings, the main contributions of our work include: (i) we formulate two greedy nonparametric algorithms: additive forward regression (AFR) for sparse additive models and generalized forward regression (GFR) for general multivariate regression models. Both of them can simultaneously conduct estimation and variable selection in high dimensions. (ii) We present theoretical results for AFR using specific smoothers. (iii) We report thorough numerical results on both simulated and real-world datasets to demonstrate the superior performance of these two methods over the state-of-the-art competitors, including LASSO, SpAM, and an adaptive parametric forward-backward algorithm called Foba [22].

The rest of this paper is organized as follows: in the next section we review the basic problem formulation and notations. In Section 3 we present the AFR algorithm, in section 4, we present the GFR algorithm. Some theoretical results are given in Section 5. In Section 6 we present numerical results on both simulated and real datasets, followed by a concluding section at the end.

## 2   Sparse Nonparametric Learning in High Dimensions

We begin by introducing some notations. Assuming $n$ data points $\left\{(X^i, Y^i)\right\}_{i=1}^n$ are observed from a high dimensional regression model

$$Y^i = m(X^i) + \epsilon^i, \ \ \epsilon^i \sim N(0, \sigma^2) \ \ i = 1, \ldots, n, \tag{1}$$

where $X^i = (X_1^i, \ldots, X_p^i)^T \in \mathbf{R}^p$ is a $p$-dimensional design point, $m : \mathbf{R}^p \to \mathbf{R}$ is an unknown smooth mean function. Here we assume $m$ lies in a $p$-dimensional second order Sobolev ball with finite radius. In the sequel, we denote the response vector $(Y^1, \ldots, Y^n)^T$ by $Y$ and the vector $(X_j^1, \ldots, X_j^n)^T$ by $X_j$ for $1 \le j \le p$.

We assume $m$ is *functional sparse*, i.e. there exists an index set $S \subset \{1, \ldots, p\}$, such that

$$(\text{General}) \quad m(x) = m(x_S), \tag{2}$$

where $|S| = r \ll p$ and $x_S$ denotes the sub-vector of $x$ with elements indexed by $S$.

Sometimes, the function $m$ can be assumed to have more structures to obtain a better estimation result. The most popular one is additivity assumption [10]. In this case, $m$ decomposes into the sum of $r$ univariate functions $\{m_j\}_{j \in S}$:

$$(\text{Additive}) \quad m(x) = \alpha + \sum\nolimits_{j \in S} m_j(x_j), \tag{3}$$

where each component function $m_j$ is assumed to lie in a second order Sobolev ball with finite radius so that each element in the space is smooth enough. For the sake of identifiability, we also assume $\mathbf{E}m_j(X_j) = 0$ for $j = 1, \ldots, p$, where the expectation is taken with respect to the marginal distribution of $X_j$.

Given the models in (2) or (3), we have two tasks: *function estimation* and *variable selection*. For the first task, we try to find an estimate $\widehat{m}$, such that $\|\widehat{m} - m\| \to 0$ as $n$ goes to infinity, where $\|\cdot\|$ is some function norm. For the second task, we try to find an estimate $\widehat{S}$, which is an index set of variables, such that $\mathbf{P}\left(\widehat{S} = S\right) \to 1$ as $n$ goes to infinity.

## 3   Additive Forward Regression

In this section, we assume the true model is additive, i.e. $m(x) = \alpha + \sum_{j \in S} m_j(x_j)$. In general, if the true index set for the relevant variables is known, the backfitting algorithm can be directly

applied to estimate $\widehat{m}$ [10]. It is essentially a Gauss-Seidel iteration for solving a set of normal equations in a function space. In particular, we denote the estimates on the $j$th variable $X_j$ to be $\widehat{m}_j \equiv (\widehat{m}_j(X_j^1), \ldots, \widehat{m}_j(X_j^n))^T \in \mathbf{R}^n$. Then $\widehat{m}_j$ can be estimated by regressing the partial residual vector $R_j = Y - \alpha - \sum_{k \neq j} \widehat{m}_k$ on the variable $X_j$. This can be calculated by $\widehat{m}_j = \mathcal{S}_j R_j$, where $\mathcal{S}_j : \mathbf{R}^n \to \mathbf{R}^n$ is a smoothing matrix, which only depends on $X^1, \ldots, X^n$ but not on $Y$. Once $\widehat{m}_j$ is updated, the algorithm holds it fixed and repeats this process by cycling through each variable until convergence. Under mild conditions on the smoothing matrices $\mathcal{S}_1, \ldots, \mathcal{S}_p$, the backfitting algorithm is a first order algorithm that guarantees to converge [5] and achieves the minimax rate of convergence as if only estimating a univariate function. However, for sparse learning problems, since the true index set is unknown, the backfitting algorithm no longer works due to the uncontrolled estimation variance.

By extending the idea of the orthogonal matching pursuit to sparse additive models, we design a forward greedy algorithm called the *additive forward regression* (AFR), which only involves a few variables in each iteration. Under this framework, we only need to conduct the backfitting algorithm on a small set of variables. Thus the variance can be well controlled. The algorithm is described in Figure 1, where we use $\langle \cdot, \cdot \rangle_n$ to denote the inner product of two vectors.

---

**Input:** $\left\{ (X^i, Y^i) \right\}_{i=1}^n$ and $\eta > 0$
    let $\mathcal{A}^{(0)} = \emptyset$, $\alpha = \sum_{i=1}^n Y^i / n$ and the residual $R^{(0)} = Y - \alpha$
    **for** $k = 1, 2, 3, \ldots$
        for each $j \notin \mathcal{A}^{(k-1)}$, estimate $\widehat{m}_j$ by smoothing: $\widehat{m}_j = \mathcal{S}_j R^{(k-1)}$
        let $j^{(k)} = \underset{j \notin \mathcal{A}^{(k-1)}}{\arg\max} |\langle \widehat{m}_j, R^{(k-1)} \rangle_n|$
        let $\mathcal{A}^{(k)} = \mathcal{A}^{(k-1)} \cup j^{(k)}$
        estimate $\mathcal{M}^{(k)} = \{ m_j : j \in \mathcal{A}^{(k)} \}$ by the backfitting algorithm
        compute the residual $R^{(k)} = Y - \alpha - \sum_{m_j \in \mathcal{M}^{(k)}} m_j(X_j)$
        **if** $(\|R^{(k-1)}\|_2^2 - \|R^{(k)}\|_2^2)/n \leq \eta$
          $k = k - 1$
          **break**
        **end if**
    **end for**
**Output:** selected variables $\mathcal{A}^{(k)}$ and estimated component functions $\mathcal{M}^{(k)} = \{ m_j : j \in \mathcal{A}^{(k)} \}$

---

Figure 1: THE ADDITIVE FORWARD REGRESSION ALGORITHM

The algorithm uses an active set $\mathcal{A}$ to index the variables included in the model during each iteration and then performs a full optimization over all "active" variables via the backfitting algorithm. The main advantage of this algorithm is that during each iteration, the model inference is conducted in low dimensions and thus avoids the curse of dimensionality. The stopping criterion is controlled by a predefined parameter $\eta$ which is equivalent to the regularization tuning parameter in convex regularization methods. Other stopping criteria, such as the maximum number of steps, can also be adopted. In practice, we always recommend to use data-dependent technique, such as cross-validation, to automatically tune this parameter.

Moreover, the smoothing matrix $\mathcal{S}_j$ can be fairly general, e.g. univariate local linear smoothers as described below, kernel smoothers or spline smoothers [21], etc.

## 4 Generalized Forward Regression

This section only assume $m(x)$ to be functional sparse, i.e. $m(x) = m(x_S)$, without restricting the model to be additive. In this case, to find a good estimate $\widehat{m}$ becomes more challenging.

To estimate the general multivariate mean function $m(x)$, one of the most popular methods is the local linear regression: given an evaluation point $x = (x_1, \ldots, x_p)^T$, the estimate $\widehat{m}(x)$ is the

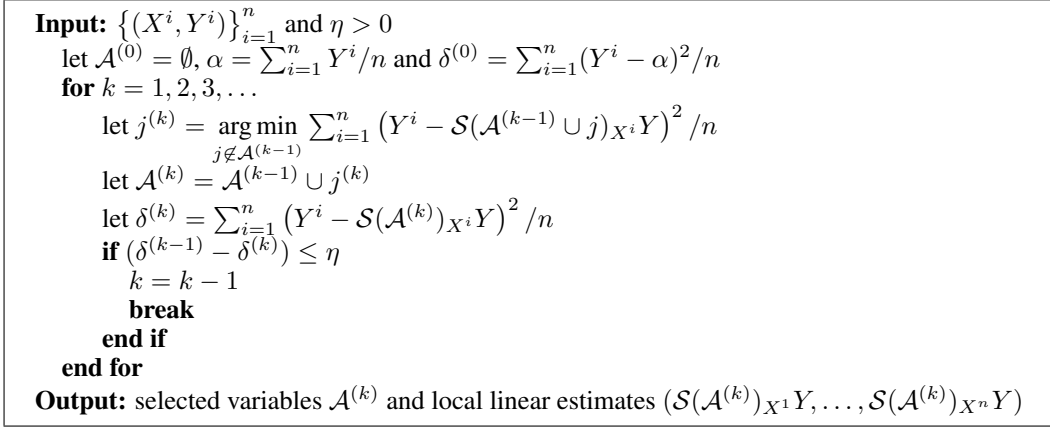

**Input:** $\left\{(X^i, Y^i)\right\}_{i=1}^{n}$ and $\eta > 0$
    let $\mathcal{A}^{(0)} = \emptyset$, $\alpha = \sum_{i=1}^{n} Y^i/n$ and $\delta^{(0)} = \sum_{i=1}^{n}(Y^i - \alpha)^2/n$
    **for** $k = 1, 2, 3, \ldots$
        let $j^{(k)} = \underset{j \notin \mathcal{A}^{(k-1)}}{\arg\min} \sum_{i=1}^{n} \left(Y^i - \mathcal{S}(\mathcal{A}^{(k-1)} \cup j)_{X^i} Y\right)^2/n$
        let $\mathcal{A}^{(k)} = \mathcal{A}^{(k-1)} \cup j^{(k)}$
        let $\delta^{(k)} = \sum_{i=1}^{n} \left(Y^i - \mathcal{S}(\mathcal{A}^{(k)})_{X^i} Y\right)^2/n$
        **if** $\left(\delta^{(k-1)} - \delta^{(k)}\right) \leq \eta$
            $k = k - 1$
            **break**
        **end if**
    **end for**
**Output:** selected variables $\mathcal{A}^{(k)}$ and local linear estimates $(\mathcal{S}(\mathcal{A}^{(k)})_{X^1} Y, \ldots, \mathcal{S}(\mathcal{A}^{(k)})_{X^n} Y)$

Figure 2: THE GENERALIZED FORWARD REGRESSION ALGORITHM

solution $\widehat{\alpha}_x$ to the following locally kernel weighted least squares problem:

$$\min_{\alpha_x, \beta_x} \sum_{i=1}^{n} \left\{Y^i - \alpha_x - \beta_x^T(X^i - x)\right\}^2 \prod_{j=1}^{p} K_{h_j}(X_j^i - x_j), \tag{4}$$

where $K(\cdot)$ is a one dimensional kernel function and the kernel weight function in (4) is taken as a product kernel with the diagonal bandwidth matrix $H^{1/2} = \text{diag}\{h_1, \ldots, h_p\}$. Such a problem can be re-casted as a standard weighted least squares regression. Therefore a closed-form solution to the the local linear estimate can be explicitly given by

$$\widehat{\alpha}_x = e_1^T(X_x^T W_x X_x)^{-1} X_x^T W_x Y = \mathcal{S}_x Y,$$

where $e_1 = (1, 0, \ldots, 0)^T$ is the first canonical vector in $\mathbf{R}^{p+1}$ and

$$W_x = \text{diag}\left\{\prod_{j=1}^{p} K_{h_j}(X_j^1 - x_j), \ldots, \prod_{j=1}^{p} K_{h_j}(X_j^n - x_j)\right\}, \quad X_x = \begin{pmatrix} 1 & (X^1 - x)^T \\ \vdots & \vdots \\ 1 & (X^n - x)^T \end{pmatrix}.$$

Here, $\mathcal{S}_x$ is the local linear smoothing matrix. Note that if we constrain $\beta_x = 0$, then the local linear estimate reduces to the kernel estimate. The pointwise rate of convergence of such an estimate has been characterized in [8]: $|\widehat{m}(x) - m(x)|^2 = O_P(n^{-4/(4+p)})$, which is extremely slow when $p > 10$.

To handle the large $p$ case, we again extend the idea of the orthogonal matching pursuit to this setting. For an index subset $\mathcal{A} \subset \{1, \ldots, p\}$ and the evaluation point $x$, the local linear smoother restricted on $\mathcal{A}$ is denoted as $\mathcal{S}(\mathcal{A})$ and

$$\mathcal{S}(\mathcal{A})_x = e_1^T \left(X(\mathcal{A})_x^T W(\mathcal{A})_x X(\mathcal{A})_x\right)^{-1} X(\mathcal{A})_x^T W(\mathcal{A})_x,$$

where $W(\mathcal{A})_x$ is a diagonal matrix whose diagonal entries are the product of univariate kernels over the set $\mathcal{A}$ and $X(\mathcal{A})_x$ is a submatrix of $X_x$ that only contains the columns indexed by $\mathcal{A}$.

Given these definitions, the *generalized forward regression* (GFR) algorithm is described in Figure 2. Similar to AFR, GFR also uses an active set $\mathcal{A}$ to index the variables included in the model. Such mechanism allows all the statistical inference to be conducted only in low-dimensional spaces. The GFR algorithm using the multivariate local linear smoother can be computationally heavy for very high dimensional problems. However, GFR is a generic framework and can be equipped with arbitrary multivariate smoothers, e.g. kernel/Nearest Neighbor/spline smoothers. These smoothers lead to much better scalability. The only reason we use the local linear smoother as an illustrative example in this paper is due to its popularity and potential advantage on correcting the boundary bias.

## 5 Theoretical Properties

In this section, we provide the theoretical properties of the additive forward regression estimates using the spline smoother. Due to the asymptotic equivalence of the spline smoother and the local linear smoother [18], we deduce that these results should also hold for the local linear smoother. Our main result in Theorem 1 says when using the spline smoother with certain truncation rate to implement AFR algorithm, the resulting estimator is consistent with a certain rate. When the underlying true component functions do not go to zeroes too fast, we also achieve variable selection consistency. Our analysis relies heavily on [3]. A similar analysis has also been reported in the technical report version of [16].

**Theorem 1.** *Assuming there exists some $\xi > 0$ which can be arbitrarily large, such that $p = O(n^\xi)$. For $\forall j \in \{1, \ldots, p\}$, we assume $m_j$ lies in a second-order Sobolev ball with finite radius, and $m = \alpha + \sum_{j=1}^p m_j$. For the additive forward regression algorithm using the spline smoother with a truncation rate at $n^{1/4}$, after $(n/\log n)^{1/2}$ steps, we obtain that*

$$\|m - \widehat{m}\|^2 = O_P\left(\sqrt{\frac{\log n}{n}}\right). \tag{5}$$

*Furthermore, if we also assume $\min_{j \in S} \|m_j\| = \Omega\left(\left(\frac{\log n}{n}\right)^{1/4}\right)$, then $\mathbf{P}\left(\widehat{S} = S\right) \to 1$ as $n$ goes to infinity. Here, $\widehat{S}$ is the index set for nonzero component functions in $\widehat{m}$.*

The rate for $\|\widehat{m} - m\|^2$ obtained from Theorem 1 is only $O(n^{-1/2})$, which is slower than the minimax rate $O(n^{-4/5})$. This is mainly an artifact of our analysis instead of a drawback of the additive forward regression algorithm. In fact, if we perform a basis expansion for each component function to first cast the problem to be a finite dimensional linear model with group structure, under some more stringent smallest eigenvalue conditions on the augmented design as in [23], we can show that AFR using spline smoothers can actually achieves the minimax rate $O(n^{-4/5})$ up to a logarithmic factor. A detailed treatment will be reported in a follow up paper.

**Sketch of Proof**: We first describe an algorithm called *group orthogonal greedy algorithm* (GOGA), which solves a noiseless function approximation problem in a direct-sum Hilbert space. AFR can then be viewed as an empirical realization of such an "ideal" algorithm.

GOGA is a group extension of the orthogonal greedy algorithm (OGA) in [3]. For $j = 1, \ldots, p$, let $\mathcal{H}_j$ be a Hilbert space of continuous functions with a Hamel basis $\mathcal{D}_j$. Then for a function $m$ in the direct-sum Hilbert space $\mathcal{H} = \mathcal{H}_1 + \mathcal{H}_2 + \ldots + \mathcal{H}_p$, we want to approximate $m$ using the union of many truncated bases $\mathcal{D} = \mathcal{D}'_1 \cup \ldots \cup \mathcal{D}'_p$, where for all $j$, $\mathcal{D}'_j \subset \mathcal{D}_j$.

We equip an inner product $\langle \cdot, \cdot \rangle$ on $\mathcal{H}$: $\forall f, g \in \mathcal{H}$, $\langle f, g \rangle = \int f(X)g(X)dP_X$ where $P_X$ is the marginal distribution for $X$. Let $\|\cdot\|$ be the norm induced by the inner product $\langle \cdot, \cdot \rangle$ on $\mathcal{H}$. GOGA begins by setting $m^{(0)} = 0$, and then recursively defines the approximant $m^{(k)}$ based on $m^{(k-1)}$ and its residual $r^{(k-1)} \equiv m - m^{(k-1)}$. More specifically: we proceed as the following: define $f_j^{(k)}$ to be the projection of $r^{(k-1)}$ onto the truncated basis $\mathcal{D}'_j$, i.e. $f_j^{(k)} = \arg\min_{g \in \mathcal{D}'_j} \|r^{(k-1)} - g\|^2$. We calculate $j^{(k)}$ as $j^{(k)} = \arg\max_j |\langle r^{(k-1)}, f_j^{(k)} \rangle|$. $m^{(k)}$ can then be calculated by projecting $m$ onto the additive function space generated by $\mathcal{A}^{(k)} = \mathcal{D}'_{j^{(1)}} + \cdots + \mathcal{D}'_{j^{(k)}}$:

$$\widehat{m}^{(k)} = \arg\min_{g \in \text{span}(\mathcal{A}^{(k)})} \|m - g\|^2.$$

AFR using regression splines is exactly GOGA when there is no noise. For noisy samples, we replace the unknown function $m$ by its $n$-dimensional output vector $Y$, and replace the inner product $\langle \cdot, \cdot \rangle$ by $\langle \cdot, \cdot \rangle_n$, which is defined as $\langle f, g \rangle_n = \frac{1}{n}\sum_{i=1}^n f(X^i)g(X^i)$. The projection of the current residual vector onto each dictionary $\mathcal{D}'_j$ is replaced by the corresponding nonparametric smoothers.

Considering any function $m \in \mathcal{H}$, we proceed in the same way as in [3], but replacing the OGA arguments in their analysis by those of GOGA. The desired results of the theorem follow from a simple argument on bounding the random random covering number of spline spaces.

# 6 Experimental Results

In this section, we present numerical results for AFR and GFR applied to both synthetic and real data. The main conclusion is that, in many cases, their performance on both function estimation and variable selection can clearly outperform those of LASSO, Foba, and SpAM. For all the reported experiments, we use local linear smoothers to implement AFR and GFR. The results for other smoothers, such as smoothing splines, are similar. Note that different bandwidth parameters will have big effects on the performances of local linear smoothers. Our experiments simply use the plug-in bandwidths according to [8] and set the bandwidth for each variable to be the same. For AFR, the bandwidth $h$ is set to be $1.06n^{-1/5}$ and for GFR, the bandwidth is varying over each iteration such that $h = 1.06n^{-1/(4+|\mathcal{A}|)}$, where $|\mathcal{A}|$ is the size of the current active set.

For an estimate $\widehat{m}$, the estimation performance for the synthetic data is measured by the mean square error (MSE), which is defined as $\mathrm{MSE}(\widehat{m}) = \frac{1}{n}\sum_{i=1}^{n}\left(m(X^i) - \widehat{m}(X^i)\right)^2$. For the real data, since we do not know the true function $m(x)$, we approximate the mean squared error using 5-fold cross-validation scores.

## 6.1 The Synthetic Data

For the synthetic data experiments, we consider the *compound symmetry* covariance structure of the design matrix $X \in \mathbb{R}^{n \times p}$ with $n = 400$ and $p = 20$. Each dimension $X_j$ is generated according to

$$X_j = \frac{W_j + tU}{1 + t}, \quad j = 1, \ldots, p,$$

where $W_1, \ldots, W_p$ and $U$ are i.i.d. sampled from Uniform(0,1). Therefore the correlation between $X_j$ and $X_k$ is $t^2/(1 + t^2)$ for $j \neq k$. We assume the true regression functions have $r = 4$ relevant variables:

$$Y = m(X) + \epsilon = m(X_1, \ldots, X_4) + \epsilon. \tag{6}$$

To evaluate the variable selection performance of different methods, we generate 50 designs and 50 trials for each design. For each trial, we run the greedy forward algorithm $r$ steps. If all the relevant variables are included in, the variable selection task for this trial is said to be successful. We report the mean and standard deviation of the success rate in variable selection for various correlation between covariates by varying the values of $t$.

We adopt some synthetic examples as in [12] and define the following four functions: $g_1(x) = x$, $g_2(x) = (2x-1)^2$, $g_3(x) = \sin(2\pi x)/(2 - \sin(2\pi x))$, and $g_4(x) = 0.1\sin(2\pi x) + 0.2\cos(2\pi x) + 0.3\sin^2(2\pi x) + 0.4\cos^3(2\pi x) + 0.5\sin^3(2\pi x)$.

The following four regression models are studied. The first model is linear; the second is additive; the third and forth are more complicated nonlinear models with at least two way interactions:

$$\begin{aligned}
(\text{Model1}): \quad & Y^i = 2X_1^i + 3X_2^i + 4X_3^i + 5X_4^i + 2N(0,1), \ \text{ with } t = 1 \ ; \\
(\text{Model2}): \quad & Y^i = 5g_1(X_1^i) + 3g_2(X_2^i) + 4g_3(X_3^i) + 6g_4(X_4^i) + 4N(0,1), \ \text{ with } t = 1 \ ; \\
(\text{Model3}): \quad & Y^i = \exp(2X_1^i X_2^i + X_3^i) + 2X_4^i + N(0,1), \ \text{ with } t = 0.5 \ ; \\
(\text{Model4}): \quad & Y^i = \sum_{j=1}^{4} g_j(X_j^i) + g_1(X_3^i X_4^i) + g_2((X_1^i + X_3^i)/2) + g_3(X_1^i X_2^i) + N(0,1) \\
& \text{with } t = 0.5.
\end{aligned}$$

Compared with LASSO, Foba, and SpAM, the estimation performance using MSE as evaluation criterion is presented in Figure 3. And Table 1 shows the rate of success for variable selection of these models with different correlations controlled by $t$.

From Figure 3, we see that AFR and GFR methods provide very good estimates for the underlying true regression functions as compared to others. Firstly, LASSO and SpAM perform very poorly when the selected model is very sparse. This is because they are convex regularization based approaches: to obtain a very sparse model, they induce very large estimation bias. On the other hand, the greedy pursuit based methods like Foba, AFR and GFR do not suffer from such a problem. Secondly, when the true model is linear, all methods perform similarly. For the nonlinear true regression

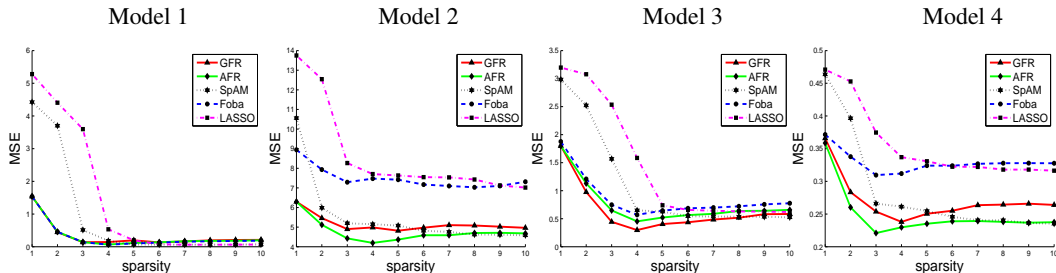

Figure 3: Performance of the different algorithms on synthetic data: MSE versus sparsity level

function, AFR, GFR and SpAM outperform LASSO and Foba. It is expectable since LASSO and Foba are based on linear assumptions. Furthermore, we notice that when the true model is additive (Model 2) or nearly additive (Model 4), AFR performs the best. However, for the non-additive general multivariate regression function (Model 3), GFR performs the best. For all examples, when more and more irrelevant variables are included in the model, SpAM has a better generalization performance due to the regularization effect.

Table 1: Comparison of variable selection

| Model 1 | LASSO(sd) | Foba | SpAM | AFR | GFR |
|---|---|---|---|---|---|
| $t = 0$ | 1.000 (0.0000) | 1.000 (0.0000) | 0.999 (0.0028) | 0.999 (0.0039) | 0.990 (0.0229) |
| $t = 1$ | 0.879 (0.0667) | 0.882 (0.0557) | 0.683 (0.1805) | 0.879 (0.0525) | 0.839 (0.0707) |
| $t = 2$ | 0.559 (0.0913) | 0.553 (0.0777) | 0.190 (0.1815) | 0.564 (0.0739) | 0.515 (0.0869) |

| Model 2 | LASSO(sd) | Foba | SpAM | AFR | GFR |
|---|---|---|---|---|---|
| $t = 0$ | 0.062 (0.0711) | 0.069 (0.0774) | 0.842 (0.1128) | 0.998 (0.0055) | 0.769 (0.1751) |
| $t = 1$ | 0.056 (0.0551) | 0.060 (0.0550) | 0.118 (0.0872) | 0.819 (0.1293) | 0.199 (0.2102) |
| $t = 2$ | 0.004 (0.0106) | 0.029 (0.0548) | 0.008 (0.0056) | 0.260 (0.1439) | 0.021 (0.0364) |

| Model 3 | LASSO(sd) | Foba | SpAM | AFR | GFR |
|---|---|---|---|---|---|
| $t = 0$ | 0.997 (0.0080) | 0.999 (0.0039) | 0.980 (0.1400) | 1.000 (0.0000) | 1.000 (0.0000) |
| $t = 1$ | 0.818 (0.1137) | 0.802 (0.1006) | 0.934 (0.1799) | 1.000 (0.0000) | 0.995 (0.0103) |
| $t = 2$ | 0.522 (0.1520) | 0.391 (0.1577) | 0.395 (0.3107) | 0.902 (0.1009) | 0.845 (0.1623) |

| Model 4 | LASSO(sd) | Foba | SpAM | AFR | GFR |
|---|---|---|---|---|---|
| $t = 0$ | 0.043 (0.0482) | 0.043 (0.0437) | 0.553 (0.1864) | 0.732 (0.1234) | 0.967 (0.0365) |
| $t = 0.5$ | 0.083 (0.0823) | 0.049 (0.0511) | 0.157 (0.1232) | 0.126 (0.0688) | 0.708 (0.1453) |
| $t = 1$ | 0.048 (0.0456) | 0.085 (0.0690) | 0.095 (0.0754) | 0.192 (0.0679) | 0.171 (0.1067) |

The variable selection performances of different methods in Table 1 are very similar to their estimation performances. We observe that, when correlation parameter $t$ becomes larger, the performances of all methods decrease. But SpAM is most sensitive to the correlation increase. In all models, the performance of SpAM can decrease more than 70% for the larger $t$; in contrast, AFR and GFR are more robust to the increased correlation between different covariates. Another interesting observation is on model 4. From the previous discussion, on this model, AFR achieves a better estimation performance. However, when comparing the variable selection performance, GFR is the best. This suggests that for nonparametric inference, the goals of estimation consistency and variable selection consistency might not be always coherent. Some tradeoffs might be needed to balance them.

## 6.2 The real data

In this subsection, we compare five methods on three real datasets: *Boston Housing*, *AutoMPG*, and *Ionosphere* data set [1]. *Boston Housing* contains 556 data points, with 13 features; *AutoMPG* 392 data points (we delete those with missing values), with 7 features and *Ionosphere* 351 data points, with 34 features and the binary output. We treat *Ionosphere* as a regression problem although the

response is binary. We run 10 times 5-fold cross validation on each dataset and plot the mean and standard deviation of MSE versus different sparsity levels in Figure 4.

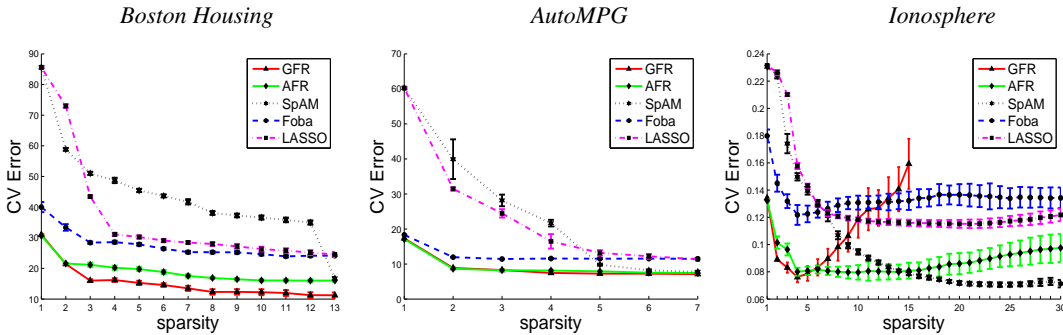

Figure 4: Performance of the different algorithms on real datasets: CV error versus sparsity level

From Figure 4, since all the error bars are tiny, we deem all the results significant. On the Boston Housing and AutoMPG datasets, the generalization performances of AFR and GFR are clearly better than LASSO, Foba, and SpAM. For all these datasets, if we prefer very sparse models, the performance of the greedy methods are much better than the convex regularization methods due to the much less bias being induced. On the Ionosphere data, we only need to run GFR up to 15 selected variables, since the generalization performance with 15 variables is already worse than the null model due to the curse of dimensionality. Both AFR and GFR on this dataset achieve the best performances when there are no more than 10 variables included; while SpAM achieves the best CV score with 25 variables. However, this is not to say that the true model is not sparse. The main reason that SpAM can achieve good generalization performance when many variables included is due to its regularization effect. We think the true model should be sparse but not additive. Similar trend among different methods has also appeared in Model 4 of previous synthetic datasets.

## 7   Conclusions and Discussions

We presented two new greedy algorithms for nonparametric regression with either additive mean functions or general multivariate regression functions. Both methods utilize the iterative forward stepwise strategy, which guarantees the model inference is always conducted in low dimensions in each iteration. These algorithms are very easy to implement and have good empirical performance on both simulated and real datasets.

One thing worthy to note is: people sometimes criticize the forward greedy algorithms since they can never have the chance to correct the errors made in the early steps. This is especially true for high dimensional linear models, which motivates the outcome of some adaptive forward-backward procedures such as Foba [22]. We addressed a similar question: Whether a forward-backward procedure also helps in the nonparametric settings? AFR and GFR can be trivially extended to be forward-backward procedures using the same way as in [22]. We conducted a comparative study to see whether the backward steps help or not. However, the backward step happens very rarely and the empirical performance is almost the same as the purely forward algorithm. This is very different from the linear model cases, where the backward step can be crucial. In summary, in the nonparametric settings, the backward ingredients will cost much more computational efforts with very tiny performance improvement. We will investigate more on this phenomenon in the near future.

A very recent research strand is to learn nonlinear models by the multiple kernel learning machinery [1, 2], another future work is to compare our methods with the multiple kernel learning approach from both theoretical and computational perspectives.

## Acknowledgements

We thank John Lafferty, Larry Wasserman, Pradeep Ravikumar, and Jamie Carbonell for very helpful discussions on this work. This research was supported in part by NSF grant CCF-0625879 and a Google Fellowship to Han Liu.

## Footnotes

[1]Available from *UCI Machine Learning Database Repository*: http:archive.ics.uci.edu/ml.

# References

[1] Francis Bach. Consistency of the group lasso and multiple kernel learning. *Journal of Machine Learning Research*, 8:1179–1225, 2008.

[2] Francis Bach. Exploring large feature spaces with hierarchical multiple kernel learning. In *Advances in Neural Information Processing Systems 21*. MIT Press, 2008.

[3] Andrew R. Barron, Albert Cohen, Wolfgang Dahmen, and Ronald A. DeVore. Approximation and learning by greedy algorithms. *The Annals of Statistics*, 36:64–94, 2008.

[4] Peter Bühlmann and Bin Yu. Sparse boosting. *Journal of Machine Learning Research*, 7:1001–1024, 2006.

[5] Andreas Buja, Trevor Hastie, and Robert Tibshirani. Linear smoothers and additive models. *The Annals of Statistics*, 17:453–510, 1989.

[6] Emmanuel Candes and Terence Tao. The dantzig selector: statistical estimation when p is much larger than n. *The Annals of Statistics*, 35:2313–2351, 2007.

[7] Scott Shaobing Chen, David L. Donoho, and Michael A. Saunders. Atomic decomposition by basis pursuit. *SIAM Journal on Scientific and Statistical Computing*, 20:33–61, 1998.

[8] Jianqing Fan and Irène Gijbels. *Local polynomial modelling and its applications*. Chapman and Hall, 1996.

[9] Jerome H. Friedman. Multivariate adaptive regression splines. *The Annals of Statistics*, 19:1–67, 1991.

[10] Trevor Hastie and Robert Tibshirani. *Generalized additive models*. Chapman & Hall Ltd., 1999.

[11] John Lafferty and Larry Wasserman. Rodeo: Sparse, greedy nonparametric regression. *The Annals of Statistics*, 36(1):28–63, 2008.

[12] Yi Lin and Hao Helen Zhang. Component selection and smoothing in multivariate nonparametric regression. *The Annals of Statistics.*, 34(5):2272–2297, 2006.

[13] Han Liu and Jian Zhang. On the estimation consistency of the group lasso and its applications. *Proceedings of the Twelfth International Conference on Artificial Intelligence and Statistics*, 2009.

[14] S. Mallat and Z. Zhang. Matching pursuit with time-frequency dictionaries. *IEEE Transactions on Signal Processing*, 41:3397–3415, 1993.

[15] Lukas Meier, Sara van de Geer, and Peter Bühlmann. High-dimensional additive modelling. *The Annals of Statistics (to appear)*, 2009.

[16] Pradeep Ravikumar, John Lafferty, Han Liu, and Larry Wasserman. Sparse additive models. *Journal of the Royal Statistical Society, Series B, Methodological*, 2009. To appear.

[17] Pradeep Ravikumar, Han Liu, John Lafferty, and Larry Wasserman. Spam: Sparse additive models. In *Advances in Neural Information Processing Systems 20*, 2007.

[18] B. W. Silverman. Spline smoothing: The equivalent variable kernel method. *The Annals of Statistics*, 12:898–916, 1984.

[19] Robert Tibshirani. Regression shrinkage and selection via the lasso. *Journal of the Royal Statistical Society, Series B, Methodological*, 58:267–288, 1996.

[20] Joel A. Tropp. Greed is good: Algorithmic results for sparse approximation. *IEEE Trans. Inform. Theory*, 50(10):2231–2241, October 2004.

[21] Grace Wahba. *Spline models for observational data*. SIAM [Society for Industrial and Applied Mathematics], 1990.

[22] Tong Zhang. Adaptive forward-backward greedy algorithm for learning sparse representations. *Technical report, Rutgers University*, 2008.

[23] Tong Zhang. On the consistency of feature selection usinggreedy least squares regression. *Journal of Machine Learning Research*, 10:555–568, 2009.

